# Penalty Decomposition Methods for Rank Minimization [*]

**Zhaosong Lu** [†]        Yong Zhang [‡]

## Abstract

In this paper we consider general rank minimization problems with rank appearing in either objective function or constraint. We first show that a class of matrix optimization problems can be solved as lower dimensional vector optimization problems. As a consequence, we establish that a class of rank minimization problems have closed form solutions. Using this result, we then propose penalty decomposition methods for general rank minimization problems. The convergence results of the PD methods have been shown in the longer version of the paper [19]. Finally, we test the performance of our methods by applying them to matrix completion and nearest low-rank correlation matrix problems. The computational results demonstrate that our methods generally outperform the existing methods in terms of solution quality and/or speed.

## 1  Introduction

In this paper we consider the following rank minimization problems:

$$\min_X \{f(X): \ \mathrm{rank}(X) \leq r, \ X \in \mathcal{X} \cap \Omega\}, \tag{1}$$

$$\min_X \{f(X) + \nu \, \mathrm{rank}(X) : X \in \mathcal{X} \cap \Omega\} \tag{2}$$

for some $r$, $\nu \geq 0$, where $\mathcal{X}$ is a closed convex set, $\Omega$ is a closed unitarily invariant set in $\Re^{m \times n}$, and $f : \Re^{m \times n} \to \Re$ is a continuously differentiable function (for the definition of unitarily invariant set, see Section 2.1). In literature, there are numerous application problems in the form of (1) or (2). For example, several well-known combinatorial optimization problems such as maximal cut (MAXCUT) and maximal stable set can be formulated as problem (1) (see, for example, [11, 1, 5]). More generally, nonconvex quadratic programming problems can also be cast into (2) (see, for example, [1]). Recently, some image recovery and machine learning problems are formulated as (1) or (2) (see, for example, [27, 31]). In addition, the problem of finding nearest low-rank correlation matrix is in the form of (1), which has important application in finance (see, for example, [4, 29, 36, 38, 25, 30, 12]).

Several approaches have recently been developed for solving problems (1) and (2) or their special cases. In particular, for those arising in combinatorial optimization (e.g., MAXCUT), one novel method is to first solve the semidefinite programming (SDP) relaxation of (1) and then obtain an approximate solution of (1) by applying some heuristics to the solution of the SDP (see, for example, [11]). Despite the remarkable success on those problems, it is not clear about the performance of this method when extended to solve more general problem (1). In addition, the nuclear norm relaxation approach has been proposed for problems (1) or (2). For example, Fazel et al. [10] considered a

---

[*]This work was supported in part by NSERC Discovery Grant.

[†]Department of Mathematics, Simon Fraser University, Burnaby, BC, V5A 1S6, Canada. (email: zhaosong@sfu.ca).

[‡]Department of Mathematics, Simon Fraser University, Burnaby, BC, V5A 1S6, Canada. (email: yza30@sfu.ca).

special case of problem (2) with $f \equiv 0$ and $\Omega = \Re^{m \times n}$. In their approach, a convex relaxation is applied to (1) or (2) by replacing the rank of $X$ by the nuclear norm of $X$ and numerous efficient methods can then be applied to solve the resulting convex problems. Recently, Recht et al. [27] showed that under some suitable conditions, such a convex relaxation is tight when $\mathcal{X}$ is an affine manifold. The quality of such a relaxation, however, remains unknown when applied to general problems (1) and (2). Additionally, for some application problems, the nuclear norm stays constant in feasible region. For example, as for nearest low-rank correlation matrix problem (see Subsection 3.2), any feasible point is a symmetric positive semidefinite matrix with all diagonal entries equal to one. For those problems, nuclear norm relaxation approach is obviously inappropriate. Finally, nonlinear programming (NLP) reformulation approach has been applied for problem (1) (see, for example, [5]). In this approach, problem (1) is cast into an NLP problem by replacing the constraint $\mathrm{rank}(X) \leq r$ by $X = UV$ where $U \in \Re^{m \times r}$ and $V \in \Re^{r \times n}$, and then numerous optimization methods can be applied to solve the resulting NLP. It is not hard to observe that such an NLP has infinitely many local minima, and moreover it can be highly nonlinear, which might be challenging for all existing numerical optimization methods for NLP. Also, it is not clear whether this approach can be applied to problem (2).

In this paper we consider general rank minimization problems (1) and (2). We first show that a class of matrix optimization problems can be solved as lower dimensional vector optimization problems. As a consequence, we establish that a class of rank minimization problems have closed form solutions. Using this result, we then propose penalty decomposition methods for general rank minimization problems in which each subproblem is solved by a block coordinate descend method. The convergence of the PD methods has been shown in the longer version of the paper [19]. Finally, we test the performance of our methods by applying them to matrix completion and nearest low-rank correlation matrix problems. The computational results demonstrate that our methods generally outperform the existing methods in terms of solution quality and/or speed.

The rest of this paper is organized as follows. In Subsection 1.1, we introduce the notation that is used throughout the paper. In Section 2, we first establish some technical results on a class of rank minimization problems and then use them to develop the penalty decomposition methods for solving problems (1) and (2). In Section 3, we conduct numerical experiments to test the performance of our penalty decomposition methods for solving matrix completion and nearest low-rank correlation matrix problems. Finally, we present some concluding remarks in Section 4.

## 1.1 Notation

In this paper, the symbol $\Re^n$ denotes the $n$-dimensional Euclidean space, and the set of all $m \times n$ matrices with real entries is denoted by $\Re^{m \times n}$. The spaces of $n \times n$ symmetric matrices will be denoted by $\mathcal{S}^n$. If $X \in \mathcal{S}^n$ is positive semidefinite, we write $X \succeq 0$. The cone of positive semidefinite matrices is denoted by $\mathcal{S}_+^n$. The Frobenius norm of a real matrix $X$ is defined as $\|X\|_F := \sqrt{\mathrm{Tr}(XX^T)}$ where $\mathrm{Tr}(\cdot)$ denotes the trace of a matrix, and the nuclear norm of $X$, denoted by $\|X\|_*$, is defined as the sum of all singular values of $X$. The rank of a matrix $X$ is denoted by $\mathrm{rank}(X)$. We denote by $I$ the identity matrix, whose dimension should be clear from the context. For a real symmetric matrix $X$, $\lambda(X)$ denotes the vector of all eigenvalues of $X$ arranged in nondecreasing order and $\Lambda(X)$ is the diagonal matrix whose $i$th diagonal entry is $\lambda_i(X)$ for all $i$. Similarly, for any $X \in \Re^{m \times n}$, $\sigma(X)$ denotes the $q$-dimensional vector consisting of all singular values of $X$ arranged in nondecreasing order, where $q = \min(m, n)$, and $\Sigma(X)$ is the $m \times n$ matrix whose $i$th diagonal entry is $\sigma_i(X)$ for all $i$ and all off-diagonal entries are 0, that is, $\Sigma_{ii}(X) = \sigma_i(X)$ for $1 \leq i \leq q$ and $\Sigma_{ij}(X) = 0$ for all $i \neq j$. We define the operator $\mathscr{D} : \Re^q \to \Re^{m \times n}$ as follows:

$$\mathscr{D}_{ij}(x) = \begin{cases} x_i & \text{if } i = j; \\ 0 & \text{otherwise} \end{cases} \qquad \forall x \in \Re^q,$$

where $q = \min(m, n)$. For any real vector, $\|\cdot\|_0$, $\|\cdot\|_1$ and $\|\cdot\|_2$ denote the cardinality (i.e., the number of nonzero entries), the standard 1-norm and the Euclidean norm of the vector, respectively.

## 2 Penalty decomposition methods

In this section, we first establish some technical results on a class of rank minimization problems. Then we propose penalty decomposition (PD) methods for solving problems (1) and (2) by using these technical results.

### 2.1 Technical results on special rank minimization

In this subsection we first show that a class of matrix optimization problems can be solved as lower dimensional vector optimization problems. As a consequence, we establish a result that a class of rank minimization problems have closed form solutions, which will be used to develop penalty decomposition methods in Subsection 2.2. The proof of the result can be found in the longer version of the paper [19]. Before proceeding, we introduce some definitions that will be used subsequently.

Let $\mathcal{U}^n$ denote the set of all unitary matrices in $\Re^{n \times n}$. A norm $\| \cdot \|$ is a *unitarily invariant norm* on $\Re^{m \times n}$ if $\|UXV\| = \|X\|$ for all $U \in \mathcal{U}^m$, $V \in \mathcal{U}^n$, $X \in \Re^{n \times n}$. More generally, a function $F : \Re^{m \times n} \to \Re$ is a *unitarily invariant function* if $F(UXV) = F(X)$ for all $U \in \mathcal{U}^m$, $V \in \mathcal{U}^n$, $X \in \Re^{m \times n}$. A set $\mathcal{X} \subseteq \Re^{m \times n}$ is a *unitarily invariant set* if

$$\{UXV : U \in \mathcal{U}^m, V \in \mathcal{U}^n, X \in \mathcal{X}\} = \mathcal{X}.$$

Similarly, a function $F : \mathcal{S}^n \to \Re$ is a *unitary similarity invariant function* if $F(UXU^T) = F(X)$ for all $U \in \mathcal{U}^n$, $X \in \mathcal{S}^n$. A set $\mathcal{X} \subseteq \mathcal{S}^n$ is a *unitary similarity invariant set* if

$$\{UXU^T : U \in \mathcal{U}^n, X \in \mathcal{X}\} = \mathcal{X}.$$

The following result establishes that a class of matrix optimization problems over a subset of $\Re^{m \times n}$ can be solved as lower dimensional vector optimization problems.

**Proposition 2.1** *Let $\| \cdot \|$ be a unitarily invariant norm on $\Re^{m \times n}$, and let $F : \Re^{m \times n} \to \Re$ be a unitarily invariant function. Suppose that $\mathcal{X} \subseteq \Re^{m \times n}$ is a unitarily invariant set. Let $A \in \Re^{m \times n}$ be given, $q = \min(m, n)$, and let $\phi$ be a non-decreasing function on $[0, \infty)$. Suppose that $U\Sigma(A)V^T$ is the singular value decomposition of A. Then, $X^* = U\mathcal{D}(x^*)V^T$ is an optimal solution of the problem*

$$\begin{aligned} \min \quad & F(X) + \phi(\|X - A\|) \\ \text{s.t.} \quad & X \in \mathcal{X}, \end{aligned} \tag{3}$$

*where $x^* \in \Re^q$ is an optimal solution of the problem*

$$\begin{aligned} \min \quad & F(\mathcal{D}(x)) + \phi(\|\mathcal{D}(x) - \Sigma(A)\|) \\ \text{s.t.} \quad & \mathcal{D}(x) \in \mathcal{X}. \end{aligned} \tag{4}$$

As some consequences of Proposition 2.1, we next state that a class of rank minimization problems on a subset of $\Re^{m \times n}$ can be solved as lower dimensional vector minimization problems.

**Corollary 2.2** *Let $\nu \geq 0$ and $A \in \Re^{m \times n}$ be given, and let $q = \min(m, n)$. Suppose that $\mathcal{X} \subseteq \Re^{m \times n}$ is a unitarily invariant set, and $U\Sigma(A)V^T$ is the singular value decomposition of A. Then, $X^* = U\mathcal{D}(x^*)V^T$ is an optimal solution of the problem*

$$\min\{\nu \operatorname{rank}(X) + \frac{1}{2}\|X - A\|_F^2 : X \in \mathcal{X}\}, \tag{5}$$

*where $x^* \in \Re^q$ is an optimal solution of the problem*

$$\min\{\nu\|x\|_0 + \frac{1}{2}\|x - \sigma(A)\|_2^2 : \mathcal{D}(x) \in \mathcal{X}\}. \tag{6}$$

**Corollary 2.3** *Let $r \geq 0$ and $A \in \Re^{m \times n}$ be given, and let $q = \min(m, n)$. Suppose that $\mathcal{X} \subseteq \Re^{m \times n}$ is a unitarily invariant set, and $U\Sigma(A)V^T$ is the singular value decomposition of A. Then, $X^* = U\mathcal{D}(x^*)V^T$ is an optimal solution of the problem*

$$\min\{\|X - A\|_F : \operatorname{rank}(X) \leq r, \ X \in \mathcal{X}\}, \tag{7}$$

*where $x^* \in \Re^q$ is an optimal solution of the problem*

$$\min\{\|x - \sigma(A)\|_2 : \|x\|_0 \leq r, \ \mathcal{D}(x) \in \mathcal{X}\}. \tag{8}$$

*Remark.* When $\mathcal{X}$ is simple enough, problems (5) and (7) have closed form solutions. In many applications, $\mathcal{X} = \{X \in \Re^{m \times n} : a \le \sigma_i(X) \le b \,\forall i\}$ for some $0 \le a < b \le \infty$. For such $\mathcal{X}$, one can see that $\mathscr{D}(x) \in \mathcal{X}$ if and only if $a \le |x_i| \le b$ for all $i$. In this case, it is not hard to observe that problems (6) and (8) have closed form solutions (see [20]). It thus follows from Corollaries 2.2 and 2.3 that problems (5) and (7) also have closed form solutions.

The following results are heavily used in [6, 22, 34] for developing algorithms for solving the nuclear norm relaxation of matrix completion problems. They can be immediately obtained from Proposition 2.1.

**Corollary 2.4** *Let $\nu \ge 0$ and $A \in \Re^{m \times n}$ be given, and let $q = \min(m, n)$. Suppose that $U\Sigma(A)V^T$ is the singular value decomposition of $A$. Then, $X^* = U\mathscr{D}(x^*)V^T$ is an optimal solution of the problem*

$$\min \nu\|X\|_* + \frac{1}{2}\|X - A\|_F^2,$$

*where $x^* \in \Re^q$ is an optimal solution of the problem*

$$\min \nu\|x\|_1 + \frac{1}{2}\|x - \sigma(A)\|_2^2.$$

**Corollary 2.5** *Let $r \ge 0$ and $A \in \Re^{m \times n}$ be given, and let $q = \min(m, n)$. Suppose that $U\Sigma(A)V^T$ is the singular value decomposition of $A$. Then, $X^* = U\mathscr{D}(x^*)V^T$ is an optimal solution of the problem*

$$\min\{\|X - A\|_F : \|X\|_* \le r\},$$

*where $x^* \in \Re^q$ is an optimal solution of the problem*

$$\min\{\|x - \sigma(A)\|_2 : \|x\|_1 \le r\}.$$

Clearly, the above results can be generalized to solve a class of matrix optimization problems over a subset of $\mathcal{S}^n$. The details can be found in the longer version of the paper [19].

## 2.2 Penalty decomposition methods for solving (1) and (2)

In this subsection, we consider the rank minimization problems (1) and (2). In particular, we first propose a penalty decomposition (PD) method for solving problem (1), and then extend it to solve problem (2) at end of this subsection. Throughout this subsection, we make the following assumption for problems (1) and (2).

**Assumption 1** *Problems (1) and (2) are feasible, and moreover, at least a feasible solution, denoted by $X^{\mathrm{feas}}$, is known.*

Clearly, problem (1) can be equivalently reformulated as

$$\min_{X,Y}\{f(X) : X - Y = 0, \ X \in \mathcal{X}, \ Y \in \mathcal{Y}\}, \tag{9}$$

where $\mathcal{Y} := \{Y \in \Omega | \operatorname{rank}(Y) \le r\}$.

Given a penalty parameter $\varrho > 0$, the associated quadratic penalty function for (9) is defined as

$$Q_\varrho(X, Y) := f(X) + \frac{\varrho}{2}\|X - Y\|_F^2. \tag{10}$$

We now propose a PD method for solving problem (9) (or, equivalently, (1)) in which each penalty subproblem is approximately solved by a block coordinate descent (BCD) method.

**Penalty decomposition method for (9) (asymmetric matrices):**

Let $\varrho_0 > 0$, $\sigma > 1$ be given. Choose an arbitrary $Y_0^0 \in \mathcal{Y}$ and a constant $\Upsilon \ge \max\{f(X^{\mathrm{feas}}), \min_{X \in \mathcal{X}} Q_{\varrho_0}(X, Y_0^0)\}$. Set $k = 0$.

1) Set $l = 0$ and apply the BCD method to find an approximate solution $(X^k, Y^k) \in \mathcal{X} \times \mathcal{Y}$ for the penalty subproblem

$$\min\{Q_{\varrho_k}(X, Y) : X \in \mathcal{X}, \ Y \in \mathcal{Y}\} \tag{11}$$

by performing steps 1a)-1d):

1a) Solve $X_{l+1}^k \in \text{Arg}\min\limits_{X \in \mathcal{X}} Q_{\varrho_k}(X, Y_l^k)$.

1b) Solve $Y_{l+1}^k \in \text{Arg}\min\limits_{Y \in \mathcal{Y}} Q_{\varrho_k}(X_{l+1}^k, Y)$.

1c) Set $(X^k, Y^k) := (X_{l+1}^k, Y_{l+1}^k)$.

2) Set $\varrho_{k+1} := \sigma \varrho_k$.

3) If $\min\limits_{X \in \mathcal{X}} Q_{\varrho_{k+1}}(X, Y^k) > \Upsilon$, set $Y_0^{k+1} := X^{\text{feas}}$. Otherwise, set $Y_0^{k+1} := Y^k$.

4) Set $k \leftarrow k + 1$ and go to step 1).

**end**

*Remark.* We observe that the sequence $\{Q_{\varrho_k}(X_l^k, Y_l^k)\}$ is non-increasing for any fixed $k$. Thus, in practical implementation, it is reasonable to terminate the BCD method based on the relative progress of $\{Q_{\varrho_k}(X_l^k, Y_l^k)\}$. In particular, given accuracy parameter $\epsilon_I > 0$, one can terminate the BCD method if

$$\frac{|Q_{\varrho_k}(X_l^k, Y_l^k) - Q_{\varrho_k}(X_{l-1}^k, Y_{l-1}^k)|}{\max(|Q_{\varrho_k}(X_l^k, Y_l^k)|, 1)} \leq \epsilon_I. \tag{12}$$

Moreover, we can terminate the outer iterations of the above method once

$$\max_{ij} |X_{ij}^k - Y_{ij}^k| \leq \epsilon_O \tag{13}$$

for some $\epsilon_O > 0$. In addition, given that problem (11) is nonconvex, the BCD method may converge to a stationary point. To enhance the quality of approximate solutions, one may execute the BCD method multiple times starting from a suitable perturbation of the current approximate solution. In detail, at the $k$th outer iteration, let $(X^k, Y^k)$ be a current approximate solution of (11) obtained by the BCD method, and let $r_k = \text{rank}(Y^k)$. Assume that $r_k > 1$. Before starting the $(k + 1)$th outer iteration, one can apply the BCD method again starting from $Y_0^k \in \text{Arg}\min\{\|Y - Y^k\|_F : \text{rank}(Y) \leq r_k - 1\}$ (namely, a rank-one perturbation of $Y^k$) and obtain a new approximate solution $(\tilde{X}^k, \tilde{Y}^k)$ of (11). If $Q_{\varrho_k}(\tilde{X}^k, \tilde{Y}^k)$ is "sufficiently" smaller than $Q_{\varrho_k}(X^k, Y^k)$, one can set $(X^k, Y^k) := (\tilde{X}^k, \tilde{Y}^k)$ and repeat the above process. Otherwise, one can terminate the $k$th outer iteration and start the next outer iteration. Furthermore, in view of Corollary 2.3, the subproblem in step 1b) can be reduced to the problem in form of (8), which has closed form solution when $\Omega$ is simple enough. Finally, the convergence results of this PD method has been shown in the longer version of the paper [19]. Under some suitable assumptions, we have established that any accumulation point of the sequence generated by our method when applied to problem (1) is a stationary point of a nonlinear reformulation of the problem.

Before ending this section, we extend the PD method proposed above to solve problem (2). Clearly, (2) can be equivalently reformulated as

$$\min_{X,Y}\{f(X) + \nu \, \text{rank}(Y) : X - Y = 0, \ X \in \mathcal{X}, \ Y \in \Omega\}. \tag{14}$$

Given a penalty parameter $\varrho > 0$, the associated quadratic penalty function for (14) is defined as

$$P_{\varrho}(X, Y) := f(X) + \nu \, \text{rank}(Y) + \frac{\varrho}{2}\|X - Y\|_F^2. \tag{15}$$

Then we can easily adapt the PD method for solving (9) to solve (14) (or, equivalently, (2)) by setting the constant $\Upsilon \geq \max\{f(X^{\text{feas}}) + \nu \, \text{rank}(X^{\text{feas}}), \min_{X \in \mathcal{X}} P_{\varrho_0}(X, Y_0^0)\}$. In addition, the set $\mathcal{Y}$ becomes $\Omega$.

In view of Corollary 2.2, the BCD subproblem in step 1b) when applied to minimize the penalty function (15) can be reduced to the problem in form of (6), which has closed form solution when $\Omega$ is simple enough. In addition, the practical termination criteria proposed for the previous PD method can be suitably applied to this method as well. Moreover, given that problem arising in step 1) is nonconvex, the BCD method may converge to a stationary point. To enhance the quality of approximate solutions, one may apply a similar strategy as described for the previous PD method by executing the BCD method multiple times starting from a suitable perturbation of the current approximate solution. Finally, by a similar argument as in the proof of [19, Theorem 3.1], we can show that every accumulation point of the sequence $\{(X^k, Y^k)\}$ is a feasible point of (14). Nevertheless, it is not clear whether a similar convergence result as in [19, Theorem 3.1(b)] can be established due to the discontinuity and nonconvexity of the objective function of (2).

# 3 Numerical results

In this section, we conduct numerical experiments to test the performance of our penalty decomposition (PD) methods proposed in Section 2 by applying them to solve matrix completion and nearest low-rank correlation matrix problems. All computations below are performed on an Intel Xeon E5410 CPU (2.33GHz) and 8GB RAM running Red Hat Enterprise Linux (kernel 2.6.18). The codes of all the compared methods in this section are written in Matlab.

## 3.1 Matrix completion problem

In this subsection, we apply our PD method proposed in Section 2 to the matrix completion problem, which has numerous applications in control and systems theory, image recovery and data mining (see, for example, [33, 24, 9, 16]). It can be formulated as

$$
\begin{aligned}
\min_{X \in \Re^{m \times n}} \quad & \mathrm{rank}(X) \\
\mathrm{s.t.} \quad & X_{ij} = M_{ij}, \ (i,j) \in \Theta,
\end{aligned}
\tag{16}
$$

where $M \in \Re^{m \times n}$ and $\Theta$ is a subset of index pairs $(i,j)$. Recently, numerous methods were proposed to solve the nuclear norm relaxation or the variant of (16) (see, for example, [18, 6, 22, 8, 13, 14, 21, 23, 32, 17, 37, 35]).

It is not hard to see that problem (16) is a special case of the general rank minimization problem (2) with $f(X) \equiv 0$, $\nu = 1$, $\Omega = \Re^{m \times n}$, and $\mathcal{X} = \{X \in \Re^{m \times n} : X_{ij} = M_{ij}, \ (i,j) \in \Theta\}$. Thus, the PD method proposed in Subsection 2.2 for problem (2) can be suitably applied to (16). The implementation details of the PD method can be found in [19].

Next we conduct numerical experiments to test the performance of our PD method for solving matrix completion problem (16) on real data. In our experiment, we aim to test the performance of our PD method for solving a grayscale image inpainting problem [2]. This problem has been used in [22, 35] to test FPCA and LMaFit, respectively and we use the same scenarios as generated in [22, 35]. For an image inpainting problem, our goal is to fill the missing pixel values of the image at given pixel locations. The missing pixel positions can be either randomly distributed or not. As shown in [33, 24], this problem can be solved as a matrix completion problem if the image is of low-rank. In our test, the original $512 \times 512$ grayscale image is shown in Figure 1(a). To obtain the data for problem (16), we first apply the singular value decomposition to the original image and truncate the resulting decomposition to get an image of rank $40$ shown in Figure 1(e). Figures 1(b) and 1(c) are then constructed from Figures 1(a) and 1(e) by sampling half of their pixels uniformly at random, respectively. Figure 1(d) is generated by masking $6\%$ of the pixels of Figure 1(e) in a non-random fashion. We now apply our PD method to solve problem (16) with the data given in Figures 1(b), 1(c) and 1(d), and the resulting recovered images are presented in Figures 1(f), 1(g) and 1(h), respectively. In addition, given an approximate recovery $X^*$ for $M$, we define the relative error as

$$
\mathrm{rel\_err} := \frac{\|X^* - M\|_F}{\|M\|_F}.
$$

We observe that the relative errors of three recovered images to the original images by our method are 6.72e-2, 6.43e-2 and 6.77e-2, respectively, which are all smaller than those reported in [22, 35].

## 3.2 Nearest low-rank correlation matrix problem

In this subsection, we apply our PD method proposed in Section 2 to find the nearest low-rank correlation matrix, which has important applications in finance (see, for example, [4, 29, 36, 38, 30]). It can be formulated as

$$
\begin{aligned}
\min_{X \in S^n} \quad & \tfrac{1}{2}\|X - C\|_F^2 \\
\mathrm{s.t.} \quad & \mathrm{diag}(X) = e, \\
& \mathrm{rank}(X) \leq r, \ X \succeq 0
\end{aligned}
\tag{17}
$$

for some correlation matrix $C \in S_+^n$ and some integer $r \in [1, n]$, where $\mathrm{diag}(X)$ denotes the vector consisting of the diagonal entries of $X$ and $e$ is the all-ones vector. Recently, a few methods have been proposed for solving problem (17) (see, for example, [28, 26, 3, 25, 12, 15]).

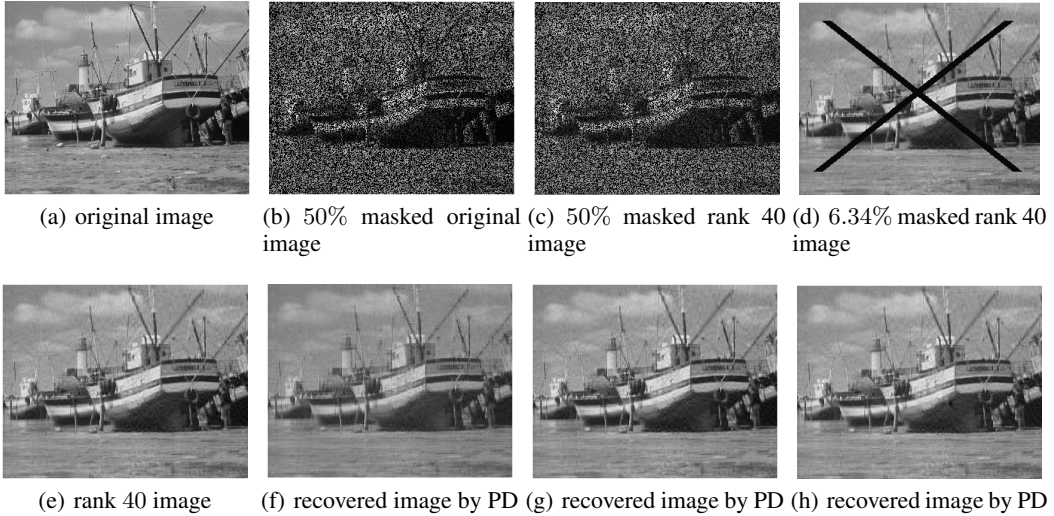

| (a) original image | (b) 50% masked original image | (c) 50% masked rank 40 image | (d) 6.34% masked rank 40 image |

| (e) rank 40 image | (f) recovered image by PD | (g) recovered image by PD | (h) recovered image by PD |

Figure 1: Image inpainting

It is not hard to see that problem (17) is a special case of the general rank constraint problem (2) with $f(X) = \frac{1}{2}\|X - C\|_F^2$, $\Omega = \mathcal{S}_+^n$, and $\mathcal{X} = \{X \in \mathcal{S}^n : \operatorname{diag}(X) = e\}$. Thus, the PD method proposed in Subsection 2.2 for problem (2) can be suitably applied to (17). The implementation details of the PD method can be found in [19].

Next we conduct numerical experiments to test the performance of our method for solving (17) on three classes of benchmark testing problems. These problems are widely used in literature (see, for example, [3, 29, 25, 15]) and their corresponding data matrices $C$ are defined as follows:

(P1) $C_{ij} = 0.5 + 0.5 \exp(-0.05|i - j|)$ for all $i$, $j$ (see [3]).

(P2) $C_{ij} = \exp(-|i - j|)$ for all $i$, $j$ (see [3]).

(P3) $C_{ij} = \text{LongCorr} + (1 - \text{LongCorr}) \exp(\kappa|i - j|)$ for all $i$, $j$, where $\text{LongCorr} = 0.6$ and $\kappa = -0.1$ (see [29]).

We first generate an instance for each (P1)-(P3) by letting 500. Then we apply our PD method and the method named as Major developed in [25] to solve problem (17) on the instances generated above. To fairly compare their performance, we choose the termination criterion for Major to be the one based on the relative error rather than the (default) absolute error. More specifically, it terminates once the relative error is less than $10^{-5}$. The computational results of both methods on the instances generated above with $r = 5, 10, \ldots, 25$ are presented in Table 1. The names of all problems are given in column one and they are labeled in the same manner as described in [15]. For example, P1n500r5 means that it corresponds to problem (P1) with $n = 500$ and $r = 5$. The results of both methods in terms of number of iterations, objective function value and CPU time are reported in columns two to seven of Table 1, respectively. We observe that the objective function values for both methods are comparable though the ones for Major are slightly better on some instances. In addition, for small $r$ (say, $r = 5$), Major generally outperforms PD in terms of speed, but PD substantially outperforms Major as $r$ gets larger (say, $r = 15$).

## 4   Concluding remarks

In this paper we proposed penalty decomposition (PD) methods for general rank minimization problems in which each subproblem is solved by a block coordinate descend method. In the longer version of the paper [20], we have showed that under some suitable assumptions any accumulation point of the sequence generated by our method when applied to the rank constrained minimization problem is a stationary point of a nonlinear reformulation of the problem. The computational results on matrix completion and nearest low-rank correlation matrix problems demonstrate that our

Table 1: Comparison of Major and PD

| Problem | Major | | | PD | | |
|---|---|---|---|---|---|---|
| | Iter | Obj | Time | Iter | Obj | Time |
| P1n500r5 | 488 | 3107.0 | 22.9 | 2514 | 3107.2 | 80.7 |
| P1n500r10 | 836 | 748.2 | 51.5 | 1220 | 748.2 | 48.4 |
| P1n500r15 | 1690 | 270.2 | 137.0 | 804 | 270.2 | 37.3 |
| P1n500r20 | 3106 | 123.4 | 329.1 | 581 | 123.4 | 31.5 |
| P1n500r25 | 5444 | 65.5 | 722.0 | 480 | 65.5 | 29.4 |
| P2n500r5 | 2126 | 24248.5 | 97.8 | 3465 | 24248.5 | 112.3 |
| P2n500r10 | 3264 | 11749.5 | 199.6 | 1965 | 11749.5 | 76.6 |
| P2n500r15 | 5061 | 7584.4 | 409.9 | 1492 | 7584.4 | 70.4 |
| P2n500r20 | 4990 | 5503.2 | 532.0 | 1216 | 5503.2 | 67.2 |
| P2n500r25 | 2995 | 4256.0 | 404.1 | 1022 | 4256.0 | 69.2 |
| P3n500r5 | 2541 | 2869.3 | 116.4 | 2739 | 2869.4 | 90.4 |
| P3n500r10 | 2357 | 981.8 | 144.2 | 1410 | 981.8 | 55.4 |
| P3n500r15 | 2989 | 446.9 | 241.9 | 923 | 446.9 | 41.6 |
| P3n500r20 | 4086 | 234.7 | 438.4 | 662 | 234.7 | 33.0 |
| P3n500r25 | 5923 | 135.9 | 788.3 | 504 | 135.9 | 29.5 |

methods generally outperform the existing methods in terms of solution quality and/or speed. More computational results of the PD method can be found in the longer version of the paper [19].

# References

[1] A. Ben-Tal and A. Nemirovski. Lectures on Modern Convex Optimization: Analysis, algorithms, Engineering Applications. *MPS-SIAM Series on Optimization*, SIAM, Philadelphia, PA, USA, 2001.

[2] M. Bertalmío, G. Sapiro, V. Caselles and V. Ballester. Image inpainting. *SIGGRAPH 2000*, New Orleans, USA, 2000.

[3] D. Brigo. A note on correlation and rank reduction. Available at *www.damianobrigo.it*, 2002.

[4] D. Brigo and F. Mercurio. Interest Rate Models: Theory and Practice. Springer-Verlag, Berlin, 2001.

[5] S. Burer, R. D. C. Monteiro, and Y. Zhang. Maximum stable set formulations and heuristics based on continuous optimization. *Math. Program.*, 94:137-166, 2002.

[6] J.-F. Cai, E. J. Candès, and Z. Shen. A singular value thresholding algorithm for matrix completion. Technical report, 2008.

[7] E. J. Candés and B. Recht. Exact matrix completion via convex optimization. *Found. Comput. Math.*, 2009.

[8] W. Dai and O. Milenkovic. SET: an algorithm for consistent matrix completion. Technical report, Department of Electrical and Computer Engineering, University of Illinois, 2009.

[9] L. Eldén. Matrix methods in data mining and pattern recognition (fundamentals of algorithms). *SIAM*, Philadelphia, PA, USA, 2009.

[10] M. Fazel, H. Hindi, and S. P. Boyd. A rank minimization heuristic with application to minimum order system approximation. *P. Amer. Contr. Conf.*, 6:4734-4739, 2001.

[11] M. X. Goemans and D. P. Williamson. .878-approximation algorithms for MAX CUT and MAX 2SAT. *Lect. Notes Comput. Sc.*, 422-431, 1994.

[12] I. Grubišić and R. Pietersz. Efficient rank reduction of correlation matrices. *Linear Algebra Appl.*, 422:629-653, 2007.

[13] R. H. Keshavan and S. Oh. A gradient descent algorithm on the Grassman manifold for matrix completion. Technical report, Department of Electrical Engineering, Stanford University, 2009.

[14] K. Lee and Y. Bresler. Admira: Atomic decomposition for minimum rank approximation. Technical report, University of Illinois, Urbana-Champaign, 2009.

[15] Q. Li and H. Qi. A sequential semismooth Newton method for the nearest low-rank correlation matrix problem. Technical report, School of Mathematics, University of Southampton, UK, 2009.

[16] Z. Liu and L. Vandenberghe. Interior-point method for nuclear norm approximation with application to system identification. *SIAM J. Matrix Anal. A.*, 31:1235-1256, 2009.

[17] Y. Liu, D. Sun, and K. C. Toh. An implementable proximal point algorithmic framework for nuclear norm minimization. Technical report, National University of Singapore, 2009.

[18] Z. Lu, R. D. C. Monteiro, and M. Yuan. Convex optimization methods for dimension reduction and coefficient estimation in Multivariate Linear Regression. Accepted in *Math. Program.*, 2008.

[19] Z. Lu and Y. Zhang. Penalty decomposition methods for rank minimization. Technical report, Department of Mathematics, Simon Fraser University, Canada, 2010.

[20] Z. Lu and Y. Zhang. Penalty decomposition methods for $l_0$ minimization. Technical report, Department of Mathematics, Simon Fraser University, Canada, 2010.

[21] R. Mazumder, T. Hastie, and R. Tibshirani. Regularization methods for learning incomplete matrices. Technical report, Stanford University, 2009.

[22] S. Ma, D. Goldfarb, and L. Chen. Fixed point and Bregman iterative methods for matrix rank minimization. To appear in *Math. Program.*, 2008.

[23] R. Meka, P. Jain and I. S. Dhillon. Guaranteed rank minimization via singular value projection. Technical report, University of Texas at Austin, 2009.

[24] T. Mrita and T. Kanade. A sequential factorization method for recovering shape and motion from image streams. *IEEE T. Pattern Anal.*, 19:858-867, 1997.

[25] R. Pietersz and I. Grubiŝić. Rank reduction of correlation matrices by majorization. *Quant. Financ.*, 4:649-662, 2004.

[26] F. Rapisarda, D. Brigo and F. Mercurio. Parametrizing correlations: a geometric interpretation. Banca IMI Working Paper, 2002 (www.fabiomercurio.it).

[27] B. Recht, M. Fazel, and P. Parrilo. Guaranteed minimum-rank solutions of linear matrix equations via nuclear norm minimization. To appear in *SIAM Rev.*, 2007.

[28] R. Rebonato. On the simultaneous calibration of multifactor lognormal interest rate models to Black volatilities and to the correlation matrix. *J. Comput. Financ.*, 2:5-27, 1999.

[29] R. Rebonato. Modern Pricing and Interest-Rate Derivatives. Princeton University Press, New Jersey, 2002.

[30] R. Rebonato. Interest-rate term-structure pricing models: a review. *P. R. Soc. Lond. A-Conta.*, 460:667-728, 2004.

[31] J. D. M. Rennie and N. Srebro. Fast maximum margin matrix factorization for collaborative prediction. In *Proceedings of the International Conference of Machine Learning*, 2005.

[32] K. Toh and S. Yun. An accelerated proximal gradient algorithm for nuclear norm regularized least squares problems. Accepted in *Pac. J. Optim.*, 2009.

[33] C. Tpmasi and T. Kanade. Shape and motion from image streams under orthography: a factorization method. *Int. J. Comput. Vision*, 9:137-154, 1992.

[34] E. van den Berg and M. P. Friedlander. Sparse optimization with least-squares constraints. Technical Report, University of British Columbia, Vancouver, 2010.

[35] Z. Wen, W. Yin, and Y. Zhang. Solving a low-rank factorization model for matrix completion by a nonlinear successive over-relaxation algorithm. Technical report, Department of Computational and Applied Mathematics, Rice University, 2010.

[36] L. Wu. Fast at-the-money calibration of the LIBOR market model using Lagrangian multipliers. *J. Comput. Financ.*, 6:39-77, 2003.

[37] J. Yang and X. Yuan. An inexact alternating direction method for trace norm regularized least squares problem. Technical report, Department of Mathematics, Nanjing University, China, 2010.

[38] Z. Zhang and L. Wu. Optimal low-rank approximation to a correlation matrix. *Linear Algebra Appl.*, 364:161-187, 2003.

